# Word Space

**Hinrich Schütze**
Center for the Study of Language and Information
Ventura Hall
Stanford, CA 94305-4115

## Abstract

Representations for semantic information about words are necessary for many applications of neural networks in natural language processing. This paper describes an efficient, corpus-based method for inducing distributed semantic representations for a large number of words (50,000) from lexical coccurrence statistics by means of a large-scale linear regression. The representations are successfully applied to word sense disambiguation using a nearest neighbor method.

## 1  Introduction

Many tasks in natural language processing require access to semantic information about lexical items and text segments. For example, a system processing the sound sequence: /rékənaisbí:tʃ/ needs to know the topic of the discourse in order to decide which of the plausible hypotheses for analysis is the right one: e.g. "wreck a nice beach" or "recognize speech". Similarly, a mail filtering program has to know the topical significance of words to do its job properly.

Traditional semantic representations are ill-suited for artificial neural networks since they presume a varying number of elements in representations for different words which is incompatible with a fixed input window. Their localist nature also poses problems because semantic similarity (for example between *dog* and *cat*) may be hidden in inheritance hierarchies and complicated feature structures. Neural networks perform best when similarity of targets corresponds to similarity of inputs; traditional symbolic representations do not have this property. Microfeatures have been widely used to overcome these problems. However, microfeature representa-

tions have to be encoded by hand and don't scale up to large vocabularies.

This paper presents an efficient method for deriving vector representations for words from lexical cooccurrence counts in a large text corpus. Proximity of vectors in the space (measured by the normalized correlation coefficient) corresponds to semantic similarity. Lexical cooccurrence can be easily measured. However, for a vocabulary of 50,000 words, there are 2,500,000,000 possible cooccurrence counts to keep track of. While many of these are zero, the number of non-zero counts is still huge. On the other hand, in any document collection most of these counts are small and therefore unreliable. Therefore, **letter fourgrams** are used here to bootstrap the representations. Cooccurrence statistics are collected for 5,000 selected fourgrams. Since each of the 5000 fourgrams is frequent, counts are more reliable than cooccurrence counts for rare words. The 5000-by-5000 matrix used for this purpose is manageable. A vector for a lexical item is computed as the sum of fourgram vectors that occur close to it in the text. This process of **confusion** yields representations of words that are fine-grained enough to reflect semantic differences between the various case and inflectional forms a word may have in the corpus.

The paper is organized as follows. Section 2 discusses related work. Section 3 describes the derivation of the vector representations. Section 4 performs an evaluation. The final section concludes.

## 2   Related Work

Two kinds of semantic representations commonly used in connectionism are microfeatures (e.g. Waltz and Pollack 1985, McClelland and Kawamoto 1986) and localist schemes in which there is a separate node for each word (e.g. Cottrell 1989). Neither approach scales up well enough in its original form to be applicable to large vocabularies and a wide variety of topics. Gallant (1991), Gallant et al. (1992) present a less labor-intensive method based on microfeatures, but the features for core stems still have to be encoded by hand for each new document collection. The derivation of the *Word Space* presented here is fully automatic. It also uses feature vectors to represent words, but the features cannot be interpreted on their own. Vector similarity is the only information present in Word Space: semantically related words are close, unrelated words are distant. The emphasis on semantic similarity rather than decomposition into interpretable features is similar to Kawamoto (1988). Scholtes (1991) uses a two-dimensional Kohonen map to represent semantic similarity. While a Kohonen map can deal with non-linearities (in contrast to the singular value decomposition used below), a space of much higher dimensionality is likely to capture more of the complexity of semantic relatedness present in natural language. Scholtes' idea to use n-grams to reduce the number of initial features for the semantic representations is extended here by looking at n-gram cooccurrence statistics rather than occurrence in documents (cf. (Kimbrell 1988) for the use of n-grams in information retrieval).

An important goal of many schemes of semantic representation is to find a limited number of semantic classes (e.g. classical thesauri such as Roget's, Crouch 1990, Brown et al. 1990). Instead, a multidimensional space is constructed here, in which each word has its own individual representation. Any clustering into classes introduces artificial boundaries that cut off words from part of their semantic neighbor-

```
governor quits knights of columbus over bishop's abortion gag rule
GOVE    _QUI   NIGH    OLUM         SHOP  ABOR       _RUL
  VERN  QUIT   HTS_    LUMB         HOP_  BORT       RULE
   ERNO                                   ORTI       ULE_
   RNOR                                   RTIO
```

Figure 1: A line from the New York Times with selected fourgrams.

hood. In large classes, there will be members "from opposite sides of the class" that are only distantly related. So any class size is problematic, since words are either separated from close neighbors or lumped together with distant terms. Conversely, a multidimensional space does not make such an arbitrary classification necessary.

## 3    Derivation of the Vector Representations

**Fourgram selection.** There are about 600,000 possible fourgrams if the empty space, numbers and non-alphanumeric characters are included as "special letters". Of these, 95,000 occurred in 5 months of the New York Times. They were reduced to 5000 by first deleting all rare ones (frequency less than 1000) and then redundant and uninformative fourgrams as described below.

If there is a group of fourgrams that occurs in only one word, all but one is deleted. For instance, the fourgrams BAGH, AGHD, GHDA, HDAD tend to occur together in *Baghdad*, so three of them will be deleted. The rationale for this move is that cooccurrence information about one of the fourgrams can be fully derived from each of the others, so that an index in the matrix would be wasted if more than one of them was included. The relative frequency of one fourgram occurring after another was calculated with fivegrams. For instance, the relative frequency of AGHD following BAGH is the frequency of the fivegram BAGHD divided by the frequency of the fourgram BAGH.

Most fourgrams occur predominantly in three or four stems or words. Uninformative fourgrams are sequences such as RETI or TION that are part of so many different words (*resigned, residents, retirements, resisted, ...; abortion, desperation, construction, detention, ...*) that knowledge about coocurrence with them carries almost no semantic information. Such fourgrams are therefore useless and are deleted. Again, fivegrams were used to identify fourgrams that occurred frequently in many stems.

A set of 6290 fourgrams remained after these deletions. To reduce it to the required size of 5000, the most frequent 300 and the least frequent 990 were also deleted. Figure 1 shows a line from the New York Times and which of the 5000 selected fourgrams occurred in it.

**Computation of fourgram vectors.** The computation of word vectors described below depends on fourgram vectors that accurately reflect semantic similarity in the sense of being used to describe the same contents. Consequently, one needs to be able to compare the sets of contexts two fourgrams occur in. For this purpose, a **collocation matrix** for fourgrams was collected such that the entry $a_{i,j}$

counts the number of times that fourgram $i$ occurs at most 200 fourgrams to the left of fourgram $j$. Two columns in this matrix are similar if the contexts the corresponding fourgrams are used in are similar. The counts were determined using five months of the New York Times (June – October 1990). The resulting collocation matrix is dense: only 2% of entries are zeros, because almost any two fourgrams cooccur. Only 10% of entries are smaller than 10, so that culling small counts would not increase the sparseness of the matrix. Consequently, any computation that employs the fourgram vectors directly would be inefficient. For this reason, a singular value decomposition was performed and 97 singular values extracted (cf. Deerwester et al. 1990) using an algorithm from SVDPACK (Berry 1992). Each fourgram can then be represented by a vector of 97 real values. Since the singular value decomposition finds the best least-square approximation of the original space in 97 dimensions, two fourgram vectors will be similar if their original vectors in the collocation matrix are similar. The reduced fourgram vectors can be efficiently used for confusion as described in the following section.

**Computation of word vectors.** We can think of fourgrams as highly ambiguous terms. Therefore, they are inadequate if used directly as input to a neural net. We have to get back from fourgrams to words. For the experiment reported here, cooccurrence information was used for a second time to achieve this goal: in this case coocurrence of a target word with any of the 5000 fourgrams. For each of the selected words (see below), a context vector was computed for every position at which it occurred in the text. A context vector was defined as the sum of all defined fourgram vectors in a window of 1001 fourgrams centered around the target word. The context vectors were then normalized and summed. This sum of vectors is the vector representation of the target word. It is the **confusion** of all its uses in the corpus. More formally, if $C(w)$ is the set of positions in the corpus at which $w$ occurs and if $\varphi(f)$ is the vector representation for fourgram $f$, then the vector representation $\tau(w)$ of $w$ is defined as: (the dot stands for normalization)

$$\tau(w) = \sum_{i \in C(w)} ( \sum_{f \text{ close to } i}^{\bullet} \varphi(f))$$

The treatment of words is case-sensitive. The following terminology will be used: a *surface form* is the string of characters as it occurs in the text; a *lemma* is either lower case or upper case: all letters are lower case with the possible exception of the first; *word* is used as a case-insensitive term. So every word has exactly two lemmas. A lemma of length $n$ has up to $2^n$ surface forms. Almost every lower case lemma can be realized as an upper case surface form. But upper case lemmas are hardly ever realized as lower case surface forms.

The confusion vectors were computed for all 54366 lemmas that occurred at least 10 times in 18 months of the New York Times News Service (May 1989 – October 1990, about 50 million words). Table 1 lists the percentage of lower case and upper case lemmas, and the distribution of lemmas with respect to words.

| lemmas | number | percent |
|---|---|---|
| lower case | 32549 | 60% |
| upper case | 21817 | 40% |
| total | 54366 | 100% |

| words | number | percent |
|---|---|---|
| lower case lemma only | 23766 | 52% |
| upper case lemma only | 13034 | 29% |
| both lemmas | 8783 | 19% |
| total | 45583 | 100% |

Table 1: The distribution of lower and upper case in words and lemmas.

| word | nearest neighbors |
|---|---|
| burglar | burglars thief rob mugging stray robbing lookout chase ciate thieves |
| disable | deter intercept repel halting surveillance shield maneuvers |
| disenchantment | disenchanted sentiment resentment grudging mindful unenthusiastic |
| domestically | domestic auto/-s importers/-ed threefold inventories drastically cars |
| Dour | melodies/-dic Jazzie danceable reggae synthesizers Soul funk tunes |
| grunts | heap into ragged goose neatly pulls buzzing rake odd rough |
| kid | dad kidding mom ok buddies Mom Oh Hey hey mama |
| S.O.B. | Confessions Jill Julie biography Judith Novak Lois Learned Pulitzer |
| Ste. | dry oyster whisky hot filling rolls lean float bottle ice |
| workforce | jobs employ/-s/-ed/-ing attrition workers clerical labor hourly |
| keeping | hoping bring wiping could some would other here rest have |

Table 2: Ten random and one selected word and their nearest neighbors.

## 4    Evaluation

Table 2 shows a random sample of 10 words and their ten nearest neighbors in Word Space (or less depending on how many would fit in the table). The neighbors are listed in order of proximity to the head word. *burglar*, *disenchantment*, *kid*, and *workforce* are closely related to almost all of their nearest neighbors. The same is true for *disable*, *domestically*, and *Dour*, if we regard as the goal to come up with a characterization of semantic similarity in a corpus (as opposed to the language in general). In the New York Times, the military use of *disable* dominates, Iraq's military, oil pipelines and ships are disabled. Similarly, *domestic* usually refers to the domestic market, and only one person named *Dour* occurs in the newspaper: the Senegalese jazz musician Youssou N'Dour. So these three cases can also be counted as successes. The topic/content of *grunts* is moderately well characterized by other objects like *goose* and *rake* that one would also expect on a farm. Finally, little useful information can be extracted for *S.O.B.* and *Ste*. *S.O.B.* mainly occurs in articles about the bestseller "Confessions of an S.O.B." Since it is not used literally, its semantics don't come out very well. The neighbors of *Ste* are for the most part words associated with water, because the name of the river "Ste.-Marguerite" in Quebec (popular for salmon fishing) is the most frequent context for *Ste*. Since the significance of *Ste* depends heavily on the name it occurs in, its usefulness as a contributor of semantic information is limited, so its poor characterization should probably not be seen as problematic. The word *keeping* has been added to the table to show that the vector representations of words that can be used in a wide variety of contexts are not interesting.

Table 3 shows that it is important for many words to make a distinction between

| word | nearest neighbors |
|------|-------------------|
| pinch (.41)<br>Pinch | outs pitch Cone hitting Cary strikeout Whitehurst Teufel Dykstra mound<br>unsalted grated cloves pepper teaspoons coarsely parsley Combine cumin |
| kappa (.49)<br>Kappa | casein protein/-s synthesize liposomes recombinant enzymes amino dna<br>Phi Wesleyan graduate cum dean graduating nyu Amherst College Yale |
| roe (.54)<br>Roe | cod squid fish salmon flounder lobster haddock lobsters crab chilled<br>Wade v overturn/-ing uphold/-ing abortion Reproductive overrule |
| completion (.73)<br>completions | complete/-d/-s/-ing complex phase/-s uncompleted incomplete<br>touchdown/-s interception/-s td yardage yarder tds fumble sacked |
| ok (.60)<br>oks | d me I m wouldn t crazy you ain anymore<br>approve/-s/-d/-ing Senate Waxman bill appropriations omnibus |
| triad (.52)<br>triads | warhead/-s ballistic missile/-s ss bombers intercontinental silos<br>Triads Organized Interpol Cosa Crips gangs trafficking smuggling |

Table 3: Words for which case or inflection matter.

| word | senses | % correct | | | |
|------|--------|---|---|---|---|
| | | 1 | 2 | 3 | sum |
| *capital/s* | goods/seat of government | 96 | 92 | | 95 |
| *interest/s* | special attention/financial | 94 | 92 | | 93 |
| *motion/s* | movement/proposal | 92 | 91 | | 92 |
| *plant/s* | factory/living being | 94 | 88 | | 92 |
| *ruling* | decision/to exert control | 90 | 91 | | 90 |
| *space* | area, volume/outer space | 89 | 90 | | 90 |
| *suit/s* | legal action/garments | 94 | 95 | | 95 |
| *tank/s* | combat vehicle/receptacle | 97 | 85 | | 95 |
| *train/s* | railroad cars/to teach | 94 | 69 | | 89 |
| *vessel/s* | ship/blood vessel/hollow utensil | 93 | 91 | 86 | 92 |

Table 4: Ten disambiguation experiments using the vector representations.

lower case and upper case and between different inflections. The normalized correlation coefficient between the two case/inflectional forms of the word is indicated in each example.

**Word sense disambiguation.** Word sense disambiguation is a task that many semantic phenomena bear on and therefore well suited to evaluate the quality of semantic representations. One can use the vector representations for disambiguation in the following way. The context vector of the occurrence of an ambiguous word is defined as the sum of all word vectors ocurring in a window around it. The set of context vectors of the word in the training set can be clustered. The clustering programs used were AutoClass (Cheeseman et al. 1988) and Buckshot (Cutting et al. 1992). The clusters found (between 2 and 13) were assigned senses by inspecting a few of its members (10–20). An occurrence of an ambiguous word in the test set was then disambiguated by assigning the sense of the training cluster that was closest to its context vector. Note that this method is unsupervised in that the structure of the "sense space" is analyzed automatically by clustering. See Schütze (1992) for a more detailed description.

Table 4 lists the results for ten disambiguation experiments that were performed

using the above algorithm. Each line shows the ambiguous words, its major senses and the success rate of disambiguation for the individual senses and all major senses together. Training and test sets were taken from the New York Times newswire and were disjoint for each word. These disambiguation results are among the best reported in the literature (e.g. Yarowsky 1992). Apparently, the vector representations respect fine sense distinctions.

An interesting question is to what degree the vector representations are distributed. Using the algorithm for disambiguation described above, a set of contexts of *suit* was clustered and applied to a test text. When the first 30 dimensions were used for clustering the training set, the error rate was 9% in the test set. When only the odd dimensions were used (1,3,5,...,27,29) the error was 14%. With only the even dimensions (2,4,6,...,28,30), 13% of occurrences in the test set were misclassified. This graceful degradation indicates that the vector representations are distributed.

## 5  Discussion and Conclusion

The linear dimensionality reduction performed here could be a useful preprocessing step for other applications as well. Each of the fourgram features carries a small amount of information. Neglecting individual features degrades performance, but there are so many that they cannot be used directly as input to a neural network. The word sense disambiguation results suggest that no information is lost when only axes of variations extracted by the singular value decomposition are considered instead of the original 5000-dimensional fourgram vectors. Schütze (Forthcoming) uses the same methodology for the derivation of **syntactic** representations for words (so that verbs and nouns occupy different regions in syntactic word space). Problems in pattern recognition often have the same characteristics: uniform distribution of information over all input features or pixels and a high-dimensional input space that causes problems in training if the features are used directly. A singular value decomposition could be a useful preprocessing step for data of this nature that makes neural nets applicable to high-dimensional problems for which training would otherwise be slow if possible at all.

This paper presents Word Space, a new approach to representing semantic information about words derived from lexical cooccurrence statistics. In contrast to microfeature representations, these semantic representations can be summed for a given context to compute a representation of the topic of a text segment. It was shown that semantically related words are close in Word Space and that the vector representations can be used for word sense disambiguation. Word Space could therefore be a promising input representation for applications of neural nets in natural language processing such as information filtering or language modeling in speech recognition.

### Acknowledgements

I'm indebted to Mike Berry for SVDPACK, to NASA and RIACS for AutoClass and to the San Diego Supercomputer Center for computing resources. Thanks to Martin Kay, Julian Kupiec, Jan Pedersen, Martin Röscheisen, and Andreas Weigend for help and discussions.

# References

Berry, M. W. 1992. Large-scale sparse singular value computations. *The International Journal of Supercomputer Applications* 6(1):13–49.

Brown, P. F., V. J. D. Pietra, P. V. deSouza, J. C. Lai, and R. L. Mercer. 1990. Class-based n-gram models of natural language. Manuscript, IBM.

Cheeseman, P., J. Kelly, M. Self, J. Stutz, W. Taylor, and D. Freeman. 1988. AutoClass: A Bayesian classification system. In *Proceedings of the Fifth International Conference on Machine Learning.*

Cottrell, G. W. 1989. *A Connectionist Approach to Word Sense Disambiguation.* London: Pitman.

Crouch, C. J. 1990. An approach to the automatic construction of global thesauri. *Information Processing & Management* 26(5):629–640.

Cutting, D., D. Karger, J. Pedersen, and J. Tukey. 1992. Scatter-gather: A cluster-based approach to browsing large document collections. In *Proceedings of SIGIR'92.*

Deerwester, S., S. T. Dumais, G. W. Furnas, T. K. Landauer, and R. Harshman. 1990. Indexing by latent semantic analysis. *Journal of the American Society for Information Science* 41(6):391–407.

Gallant, S. I. 1991. A practical approach for representing context and for performing word sense disambiguation using neural networks. *Neural Computation* 3(3):293–309.

Gallant, S. I., W. R. Caid, J. Carleton, R. Hecht-Nielsen, K. P. Qing, and D. Sudbeck. 1992. HNC's matchplus system. In *Proceedings of TREC.*

Kawamoto, A. H. 1988. Distributed representations of ambiguous words and their resolution in a connectionist network. In S. L. Small, G. W. Cottrell, and M. K. Tanenhaus (Eds.), *Lexical Ambiguity Resolution: Perspectives from Psycholinguistics, Neuropsychology, and Artificial Intelligence.* San Mateo CA: Morgan Kaufmann.

Kimbrell, R. E. 1988. Searching for text? Send an N-gram! *Byte Magazine* May:297–312.

McClelland, J. L., and A. H. Kawamoto. 1986. Mechanisms of sentence processing: Assigning roles to constituents of sentences. In J. L. McClelland, D. E. Rumelhart, and the PDP Research Group (Eds.), *Parallel Distributed Processing. Explorations in the Microstructure of Cognition. Volume 2: Psychological and Biological Models,* 272–325. Cambridge MA: The MIT Press.

Scholtes, J. C. 1991. Unsupervised learning and the information retrieval problem. In *Proceedings of the International Joint Conference on Neural Networks.*

Schütze, H. 1992. Dimensions of meaning. In *Proceedings of Supercomputing '92.*

Schütze, H. Forthcoming. Sublexical tagging. In *Proceedings of the IEEE International Conference on Neural Networks.*

Waltz, D. L., and J. B. Pollack. 1985. A strongly interactive model of natural language interpretation. *Cognitive Science* 9:51–74.

Yarowsky, D. 1992. Word-sense disambiguation using statistical models of Roget's categories trained on large corpora. In *Proceedings of Coling-92.*
